# Making Latin Manuscripts Searchable using gHMM's

**Jaety Edwards  Yee Whye Teh  David Forsyth  Roger Bock  Michael Maire**
`{jaety,ywteh,daf,bock,mmaire}@cs.berkeley.edu`

**Grace Vesom**

**Department of Computer Science**
UC Berkeley
Berkeley, CA 94720

## Abstract

We describe a method that can make a scanned, handwritten mediaeval latin manuscript accessible to full text search. A generalized HMM is fitted, using transcribed latin to obtain a transition model and one example each of 22 letters to obtain an emission model. We show results for unigram, bigram and trigram models. Our method transcribes 25 pages of a manuscript of Terence with fair accuracy (75% of letters correctly transcribed). Search results are very strong; we use examples of variant spellings to demonstrate that the search respects the ink of the document. Furthermore, our model produces fair searches on a document from which we obtained no training data.

## 1. Intoduction

There are many large corpora of handwritten scanned documents, and their number is growing rapidly. Collections range from the complete works of Mark Twain to thousands of pages of zoological notes spanning two centuries. Large scale analyses of such corpora is currently very difficult, because handwriting recognition works poorly. Recently, Rath and Manmatha have demonstrated that one can use small bodies of aligned material as supervised data to train a word spotting mechanism [7]. The result can make scanned handwritten documents searchable.

Current techniques assume a closed vocabulary — one can search only for words in the training set — and search for instances of whole words. This approach is particularly unattractive for an inflected language, because individual words can take so many forms that one is unlikely to see all in the training set. Furthermore, one would like the method used to require very little aligned training data, so that it is possible to process documents written by different scribes with little overhead. Mediaeval Latin manuscripts are a natural first corpus for studying this problem, because there are many scanned manuscripts and because the handwriting is relatively regular. We expect the primary user need to be search over a large body of documents — to allow comparisons between documents — rather than transcription of a particular document (which is usually relatively easy to do by hand). Desirable features for a system are: First, that it use little or no aligned training data (an

ideal, which we believe may be attainable, is an unsupervised learning system). Second, that one can search the document for an arbitrary string (rather than, say, only complete words that appear in the training data). This would allow a user to determine whether a document contains curious or distinctive spellings, for example (figure 7).

We show that, using a statistical model based on a generalized HMM, we can search a medieval manuscript with considerable accuracy, using only one instance each of each letter in the manuscript to train the method (22 instances *in total*; Latin has no j, k, w, or z). Furthermore, our method allows fairly accurate transcription of the manuscript. We train our system on 22 glyphs taken from a a 12th century latin manuscript of Terence's Comedies (obtained from a repository of over 80 scanned medieval works maintained by Oxford University [1]). We evaluate searches using a considerable portion of this manuscript aligned by hand; we then show that fair search results are available on a different manuscript (MS. Auct. D. 2. 16, Latin Gospels with beast-headed evangelist portraits made at Landvennec, Brittany, late 9th or early 10th century, from [1]) *without change of letter templates*.

### 1.1. Previous Work

Handwriting recognition is a traditional problem, too well studied to review in detail here (see [6]). Typically, online handwriting recognition (where strokes can be recorded) works better than offline handwriting recognition. Handwritten digits can now be recognized with high accuracy [2, 5]. Handwritten amounts can be read with fair accuracy, which is significantly improved if one segments the amount into digits at the same time as one recognizes it [4, 5]. Recently several authors have proposed new techniques for search and translation in this unrestricted setting. Manmatha et al [7] introduce the technique of "word spotting," which segments text into word images, rectifies the word images, and then uses an aligned training set to learn correspondences between rectified word images and strings. The method is not suitable for a heavily inflected language, because words take so many forms. In an inflected language, the natural unit to match to is a subset of a word, rather than a whole word, implying that one should segment the text into blocks — which may be smaller than words — while recognizing. Vinciarelli et al [8] introduce a method for line by line recognition based around an HMM and quite similar to techniques used in the speech recognition community. Their method uses a window that slides along the text to obtain features; this has the difficulty that the same window is in some places too small (and so uninformative) and in others too big (and so spans more than one letter, and is confusing). Their method requires a substantial body of aligned training data, which makes it impractical for our applications. Close in spirit to our work is the approach to machine translation of Koehn and Knight [3]. They demonstrate that the statistics of unaligned corpora may provide as powerful constraints for training models as aligned bitexts.

## 2. The Model

Our models for both search and transcription are based on the generalized HMM and differ only in their choice of transition model. In an HMM, each hidden node $c_t$ emits a single evidence node $x_t$. In a generalized HMM, we allow each $c_t$ to emit a series of $x$'s whose length is itself a random variable. In our model, the hidden nodes correspond to letters and each $x_t$ is a single column of pixels. Allowing letters to emit sets of columns lets us accomodate letter templates of variable width. In particular, this means that we can unify segmenting ink into letters and recognizing blocks of ink; figure 3 shows an example of how useful this is.

### 2.1. Generating a line of text

Our hidden state consists of a character label $c$, width $w$ and vertical position $y$. The statespace of $c$ contains the characters 'a'-'z', a space ' ', and a special end state $\Omega$. Let $T_c$ be the template associated with character $c$, $T_{ch}$, $T_{cw}$ be respectively the height and width of that template, and $m$ be the height of the image.

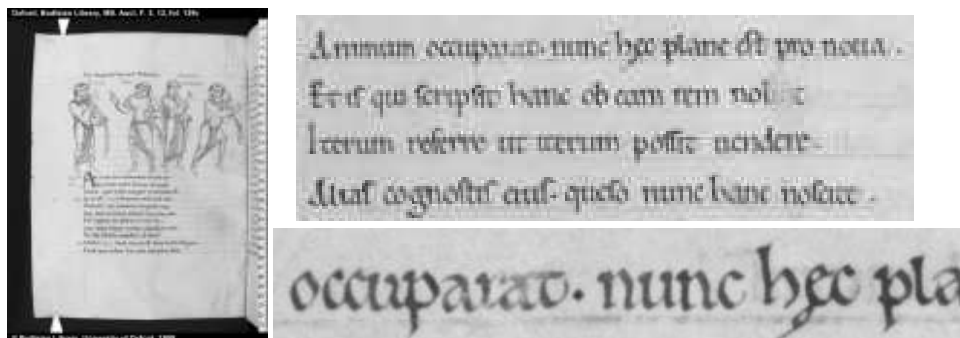

Figure 1: **Left**, *a full page of our manuscript, a 12'th century manuscript of Terence's Comedies obtained from [1].* **Top right**, *a set of lines from a page from that document and* **bottom right**, *some words in higher resolution. Note: (a) the richness of page layout; (b) the clear spacing of the lines; (c) the relatively regular handwriting.*

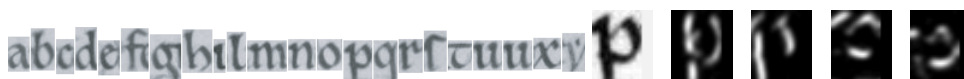

Figure 2: **Left**, *the 22 instances, one per letter, used to train our emission model. These templates are extracted by hand from the Terence document.* **Right**, *the five image channels for a single letter.*

Beginning at image column 1 (and assuming a dummy space before the first character),

- choose character $c \sim p(c|c_{-1...-n})$ (an n-gram letter model)
- choose length $w \sim \text{Uniform}(T_{cw} - k, T_{cw} + k)$ (for some small $k$)
- choose vertical position $y \sim \text{Uniform}(1, m - T_{ch})$
- *$z$,$y$ and $T_{ch}$ now define a bounding box $b$ of pixels. Let $i$ and $j$ be indexed from the top left of that bounding box.*
    - draw pixel $(i,j) \sim \mathcal{N}(T_{cij}, \sigma_{cij})$ for each pixel in $b$
    - draw all pixels above and below $b$ from background gaussian $\mathcal{N}(\mu_0, \sigma_0)$
        (See 2.2 for greater detail on pixel emission model)
- move to column $w + 1$ and repeat until we enter the end state $\Omega$.

Inference on a gHMM is a relatively straighforward business of dynamic programming. We have used unigram, bigram and trigram models, with each transition model fitted using an electronic version of Caesar's Gallic Wars, obtained from `http://www.thelatinlibrary.com`. We do not believe that the choice of author should significantly affect the fitted transition model — which is at the level of characters — but have not experimented with this point. The important matter is the emission model.

### 2.2. The Emission Model

Our emission model is as follows: Given the character $c$ and width $w$, we generate a template of the required length. Each pixel in this template becomes the mean of a gaussian which generates the corresponding pixel in the image. This template has a separate mean image for each pixel channel. The channels are assumed independent given the means.

We train the model by cutting out by hand a single instance of each letter from our corpus (figure 2). This forms the central portion of the template. Pixels above and below this

| Model | matching chars | substitutions | insertions | deletions |
|---|---|---|---|---|
| Perfect transcription | 21019 | 0 | 0 | 0 |
| unigram | 14603 | 5487 | 534 | 773 |
| bigram | 15572 | 4597 | 541 | 718 |
| trigram | 15788 | 4410 | 507 | 695 |

Table 1: *Edit distance between our transcribed Terence and the editor's version. Note the trigram model produces significantly fewer letter errors than the unigram model, but that the error rate is still a substantial 25%.*

central box are generated from a single gaussian used to model background pixels (basically white pixels). We add a third variable $y_t$ to our hidden state indicating the vertical position of the central box. However, since we are uninterested in actually recovering this variable, during inference we sum it out of the model. The width of a character is constrained to be close to the width ($t_w$) of our hand cut example by setting $p(w|c) = 0$ for $w < t_w - k$ and $w > t_w + k$. Here $k$ is a small, user defined integer. Within this range, $p(w|c)$ is distributed uniformly, larger templates are created by appending pixels from the background model to the template and smaller ones by simply removing the right $k$-most columns of the hand cut example.

For features, we generate five image representations, shown in figure 2. The first is a gray-scale version of the original color image. The second and third are generated by convolving the grayscale image with a vertical derivative of gaussian filter, separating the positive and negative components of this response, and smoothing each of these gradient images separately. The fourth and fifth are generated similarly but with a horizontal derivative of gaussian filter. We have experimented with different weightings of these 5 channels. In practice we use the gray scale channel and the horizontal gradient channels. We emphasize the horizontal pieces since these seem the more discriminative.

### 2.3. Transcription

For transcription, we model letters as coming from an $n$-gram language model, with no dependencies between words. Thus, the probability of a letter depends on the $k$ letters before it, where $k = n$ unless this would cross a word boundary in which case the history terminates at this boundary. We chose not to model word to word transition probabilities since, unlike in English, word order in Latin is highly arbitrary. This transition model is fit from a corpus of ascii encoded latin. We have experimented with unigram (i.e. uniform transition probabilities), bigram and trigram letter models. We can perform transcription by fitting the maximum likelihood path through any given line. Some results of this technique are shown in figure 3.

### 2.4. Search

For search, we rank lines by the probability that they contain our search word. We set up a finite state machine like that in figure 4. In this figure, 'bg' represents our background model for that portion of the line not generated by our search word. We can use any of the n-gram letter models described for transcription as the transition model for 'bg'. The probability that the line contains the search word is the probability that this FSM takes path 1. We use this FSM as the transition model for our gHMM, and output the posterior probability of the two arrows leading into the end state. $\epsilon_1$ and $\epsilon_2$ are user defined weights, but in practice the algorithm does not appear to be particular sensitive to the choice of these parameters. The results presented here use the unigram model.

Editorial translation *Orator ad vos venio ornatu prologi:*

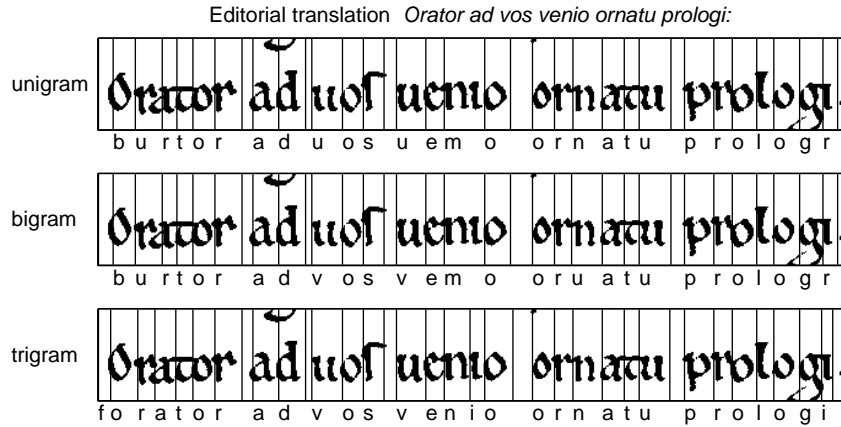

Figure 3: *We transcribe the text by finding the maximum likelihood path through the gHMM. The top line shows the standard version of the line (obtained by consensus among editors who have consulted various manuscripts; we obtained this information in electronic form from* http://www.thelatinlibrary.com*). Below, we show the line as segmented and transcribed by unigram, bigram and trigram models; the unigram and bigram models transcribe one word as "vemo", but the stronger trigram model forces the two letters to be segmented and correctly transcribes the word as "venio", illustrating the considerable benefit to be obtained by segmenting only at recognition time.*

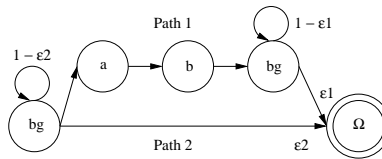

Figure 4: *The finite state machine to search for the word 'ab.' 'bg' is a place holder for the larger finite state machine defined by our language model's transition matrix.*

## 3. Results

Figure 1 shows a page from our collection. This is a scanned 12th century manuscript of Terence's Comedies, obtained from the collection at [1]. In preprocessing, we extract individual lines of text by rotating the image to various degrees and projecting the sum of the pixel values onto the y-axis. We choose the orientation whose projection vector has the lowest entropy, and then segment lines by cutting at minima of this projection.

**Transcription** is not our primary task, but methods that produce good transcriptions are going to support good searches. The gHMM can produce a surprisingly good transcription, given how little training data is used to train the emission model. We aligned an editors version of Terence with 25 pages from the manuscript by hand, and computed the edit distance between the transcribed text and the aligned text; as table 1 indicates, approximately 75% of letters are read correctly.

**Search** results are strong. We show results for two documents. The first set of results refers to the edition of Terence's Comedies, from which we took the 22 letter instances. In particular, for any given search term, our process ranks the complete set of lines. We used a hand alignment of the manuscript to determine which lines contained each term; figure 5 shows an overview of searches performed using every word that appears in the

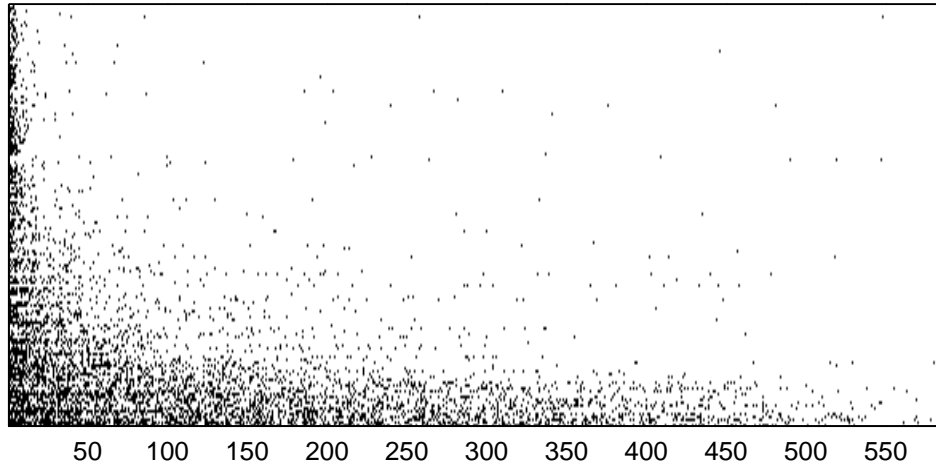

Figure 5: *Our search ranks 587 manuscript lines, with higher ranking lines more likely to contain the relevant term. This figure shows complete search results for each term that appears more than three times in the 587 lines. Each row represents the ranked search results for a term, and a black mark appears if the search term is actually in the line; a successful search will therefore appear as a row which is wholly dark to the left, and then wholly light. All 587 lines are represented. More common terms are represented by lower rows. More detailed results appear in figure 5 and figure 6; this summary figure suggests almost all searches are highly successful.*

document more than three times, in particular, showing which of the ranked set of lines actually contained the search term. For almost every search, the term appears mainly in the lines with higher rank. Figure 6 contains more detailed information for a smaller set of words. We do not score the position of a word in a line (for practical reasons).

Figure 7 demonstrates (a) that our search respects the ink of the document and (b) that for the Terence document, word positions are accurately estimated. The spelling of mediaeval documents is typically cleaned up by editors; in our manuscript, the scribe reliably spells "michi" for the standard "mihi". A search on "michi" produces many instances; a search on "mihi" produces none, because the ink doesn't have any. Notice this phenomenon also in the bottom right line of figure 7, the scribe writes "habet, ut consumat nunc cum nichil obsint doli" and the editor gives "habet, ut consumat nunc quom nil obsint doli." Figure 8 shows that searches on short strings produce many words *containing that string* as one would wish.

## 4. Discussion

We have shown that it is possible to make at least some handwritten mediaeval manuscripts accessible to full text search, without requiring an aligned text or much supervisory data. Our documents have very regular letters, and letter frequencies — which can be obtained from transcribed Latin — appear to provide so powerful a cue that relatively little detailed information about letter shapes is required. Linking letter segmentation and recognition has thoroughly beneficial effects. This suggests that the pool of manuscripts that can be made accessible in this way is large. In particular, we have used our method, trained on 22 instances of letters from one document, to search another document. Figure 9 shows the results from two searches of our second document (MS. Auct. D. 2. 16, Latin Gospels with beast-headed evangelist portraits made at Landvennec, Brittany, late 9th or early 10th century, from [1]). No information from this document was used in training at all; but letter

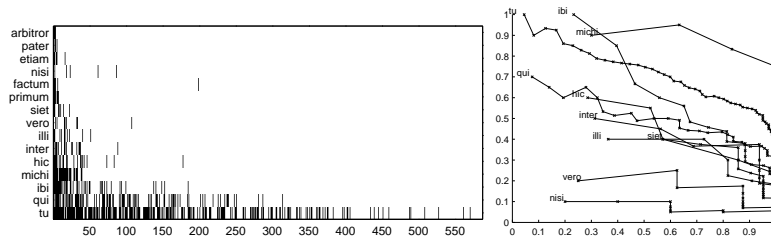

Figure 6: *On the* **left**, *search results for selected words (indicated on the leftmost column). Each row represents the ranked search results for a term, and a black mark appears if the search term is actually in the line; a successful search will therefore appear as a row which is wholly dark to the left, and then wholly light. Note only the top 300 results are represented, and that lines containing the search term are almost always at or close to the top of the search results (black marks to the left). On the* **right**, *we plot precision against recall for a set of different words by taking the top 10, 20, ... lines returned from the search, and checking them against the aligned manuscript. Note that, once all cases have been found, if the size of the pool is increased the precision will fall with 100% recall; many words work well, with most of the first 20 or so lines returned containing the search term.*

shapes are sufficiently well shared that the search is still useful.

All this suggests that one might be able to use EM to link three processes: one that clusters to determine letter shapes; one that segments letters; and one that imposes a language model. Such a system might be able to make handwritten Latin searchable with no training data.

## References

[1] Early Manuscripts at Oxford University. Bodleian library ms. auct. f. 2.13. *http://image.ox.ac.uk/.*

[2] Serge Belongie, Jitendra Malik, and Jan Puzicha. Shape matching and object recognition using shape contexts. *IEEE T. Pattern Analysis and Machine Intelligence*, 24(4):509–522, 2002.

[3] Philipp Koehn and Kevin Knight. Estimating word translation probabilities from unrelated monolingual corpora. In *Proc. of the 17th National Conf. on AI*, pages 711–715. AAAI Press / The MIT Press, 2000.

[4] Y. LeCun, L. Bottou, and Y. Bengio. Reading checks with graph transformer networks. In *International Conference on Acoustics, Speech, and Signal Processing*, volume 1, pages 151–154, Munich, 1997. IEEE.

[5] Y. Lecun, L. Bottou, Y. Bengio, and P. Haffner. Gradient-based learning applied to document recognition. *Proceedings of the IEEE*, 86(11):2278–2324, 1998.

[6] R. Plamondon and S.N. Srihari. Online and off-line handwriting recognition: a comprehensive survey. *IEEE Transactions on Pattern Analysis and Machine Intelligence*, 22(1):63–84, 2000.

[7] T. M. Rath and R. Manmatha. Word image matching using dynamic time warping. In *Proc. of the Conf. on Computer Vision and Pattern Recognition (CVPR)*, volume 2, pages 521–527, 2003.

[8] Alessandro Vinciarelli, Samy Bengio, and Horst Bunke. Offline recognition of unconstrained handwritten texts using hmms and statistical language models. *IEEE Trans. Pattern Anal. Mach. Intell.*, 26(6):709–720, 2004.

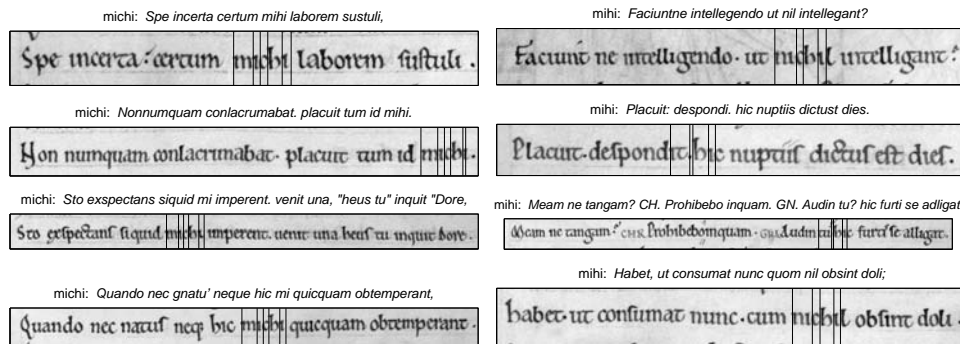

Figure 7: *The handwritten text does not fully correspond to the transcribed version; for example, scribes commonly write "michi" for the standard "mihi". Our search process reflects the ink fairly faithfully, however.* **Left** *the first four lines returned for a search on the string "michi";* **right** *the first four lines returned for a search on the string "mihi", which does not appear in the document. Note that our search process can offer scholars access to the ink in a particular document, useful for studying variations in transcription, etc.*

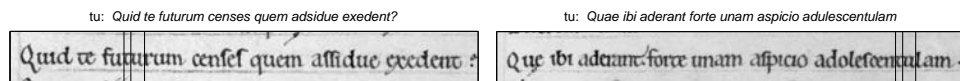

Figure 8: *Searches on short strings produce substrings of words as well as words (we show the first two lines returned from a search for "tu").*

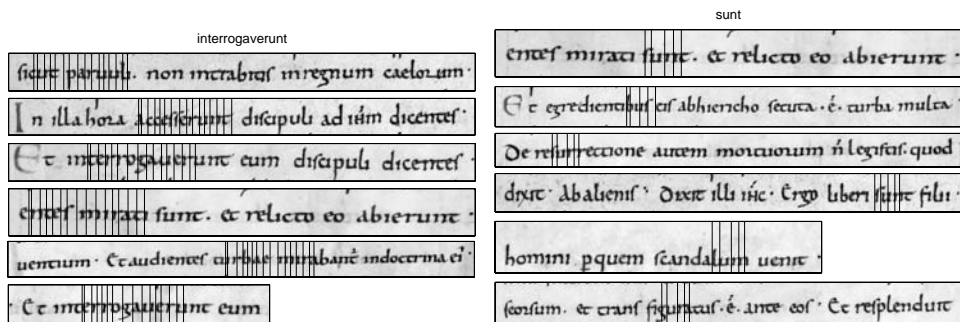

Figure 9: *The first six lines returned from the second manuscript, (MS. Auct. D. 2. 16, Latin Gospels with beast-headed evangelist portraits made at Landvennec, Brittany, late 9th or early 10th century, from [1]), in response to the queries "interrogeraverunt"* (**left***; lines three and six contain the word, which is localized largely correctly) and "sunt"* (**right***; lines one and four contain the word). We do not have aligned text, so cannot measure the recall and precision for searches on this document. The recall and precision are clearly not as good as those for the Terence document, the search is reasonably satisfactory, given that* no training information from this document was available.